# SCALING AND GENERALIZATION IN NEURAL NETWORKS: A CASE STUDY

Subutai Ahmad
Center for Complex Systems Research
University of Illinois at Urbana-Champaign
508 S. 6th St., Champaign, IL 61820

Gerald Tesauro
IBM Watson Research Center
PO Box 704
Yorktown Heights, NY 10598

## ABSTRACT

The issues of scaling and generalization have emerged as key issues in current studies of supervised learning from examples in neural networks. Questions such as how many training patterns and training cycles are needed for a problem of a given size and difficulty, how to represent the input, and how to choose useful training exemplars, are of considerable theoretical and practical importance. Several intuitive rules of thumb have been obtained from empirical studies, but as yet there are few rigorous results. In this paper we summarize a study of generalization in the simplest possible case–perceptron networks learning linearly separable functions. The task chosen was the majority function (i.e. return a 1 if a majority of the input units are on), a predicate with a number of useful properties. We find that many aspects of generalization in multilayer networks learning large, difficult tasks are reproduced in this simple domain, in which concrete numerical results and even some analytic understanding can be achieved.

## 1   INTRODUCTION

In recent years there has been a tremendous growth in the study of machines which learn. One class of learning systems which has been fairly popular is neural networks. Originally motivated by the study of the nervous system in biological organisms and as an abstract model of computation, they have since been applied to a wide variety of real-world problems (for examples see [Sejnowski and Rosenberg, 87] and [Tesauro and Sejnowski, 88]). Although the results have been encouraging, there is actually little understanding of the extensibility of the formalism. In particular, little is known of the resources required when dealing with large problems (i.e. scaling), and the abilities of networks to respond to novel situations (i.e. generalization).

The objective of this paper is to gain some insight into the relationships between three fundamental quantities under a variety of situations. In particular we are interested in the relationships between the size of the network, the number of training

instances, and the generalization that the network performs, with an emphasis on the effects of the input representation and the particular patterns present in the training set.

As a first step to a detailed understanding, we summarize a study of scaling and generalization in the simplest possible case. Using feed forward networks, the type of networks most common in the literature, we examine the majority function (return a 1 if a majority of the inputs are on), a boolean predicate with a number of useful features. By using a combination of computer simulations and analysis in the limited domain of the majority function, we obtain some concrete numerical results which provide insight into the process of generalization and which will hopefully lead to a better understanding of learning in neural networks in general.*

# 2    THE MAJORITY FUNCTION

The function we have chosen to study is the majority function, a simple predicate whose output is a 1 if and only if more than half of the input units are on. This function has a number of useful properties which facilitate a study of this type. The function has a natural appeal and can occur in several different contexts in the real-world. The problem is linearly separable (i.e. of predicate order 1 [Minsky and Papert, 69]). A version of the perceptron convergence theorem applies, so we are guaranteed that a network with one layer of weights can learn the function. Finally, when there are an odd number of input units, exactly half of the possible inputs results in an output of 1. This property tends to minimize any negative effects that may result from having too many positive or negative training examples.

# 3    METHODOLOGY

The class of networks used are *feed forward networks* [Rumelhart and McClelland, 86], a general category of networks that include perceptrons and the multi-layered networks most often used in current research. Since majority is a boolean function of predicate order 1, we use a network with no hidden units. The output function used was a sigmoid with a bias. The basic procedure consisted of three steps. First the network was initialized to some random starting weights. Next it was trained using back propagation on a set of training patterns. Finally, the performance of the network was tested on a set of random test patterns. This performance figure was used as the estimate of the network's generalization. Since there is a large amount of randomness in the procedure, most of our data are averages over several simulations.

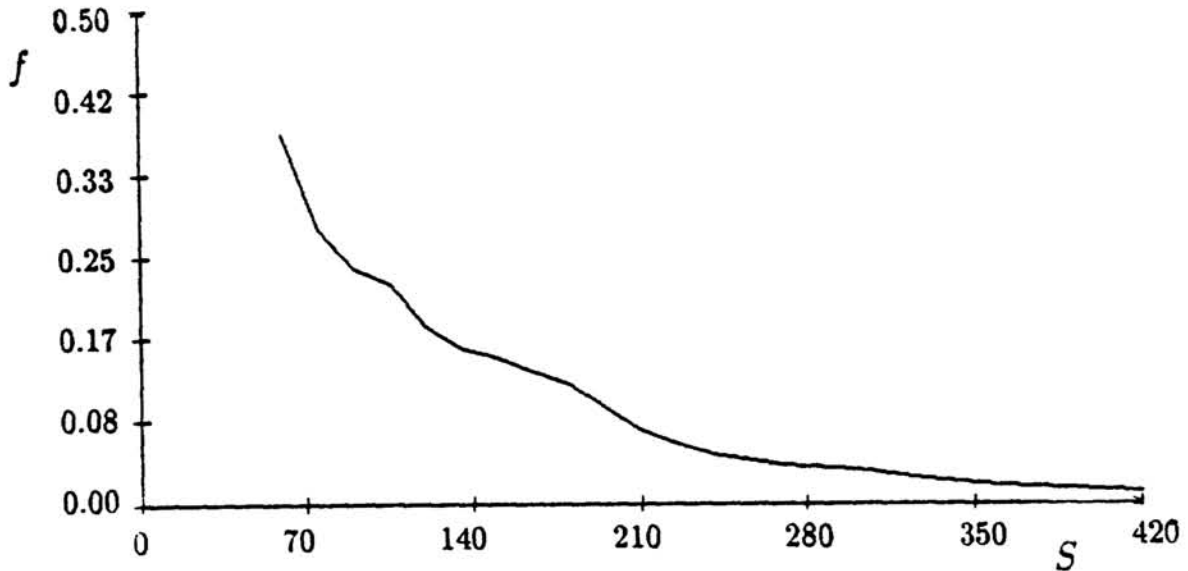

Figure 1: The average failure rate as a function of $S$. $d = 25$

**Notation.** In the following discussion, we denote $S$ to be the number of training patterns, $d$ the number of input units, and $c$ the number of cycles through the training set. Let $f$ be the *failure rate* (the fraction of misclassified training instances), and $\sigma$ be the set of training patterns.

# 4   RANDOM TRAINING PATTERNS

We first examine the failure rate as a function of $S$ and $d$. Figure 1 shows the graph of the average failure rate as a function of $S$, for a fixed input size $d = 25$. Not surprisingly we find that the failure rate decreases fairly monotonically with $S$. Our simulations show that in fact, for majority there is a well defined relationship between the failure rate and $S$. Figure 2 shows this for a network with 25 input units. The figure indicates that $\ln f$ is proportional to $S$ implying that the failure rate decreases exponentially with $S$, i.e., $f = \alpha e^{-\beta S}$. $1/\beta$ can be thought of as a characteristic training set size, corresponding to a failure rate of $\alpha/e$.

Obtaining the exact scaling relationship of $1/\beta$ was somewhat tricky. Plotting $\beta$ on a log-log plot against $d$ showed it to be close to a straight line, indicating that $1/\beta$ increases $\sim d^a$ for some constant $a$. Extracting the exponent by measuring the slope of the log-log graph turned out to be very error prone, since the data only ranged over one order of magnitude. An alternate method for obtaining the exponent is to look for a particular exponent $a$ by setting $S = \alpha d^a$. Since a linear scaling relationship is theoretically plausible, we measured the failure rate of the network

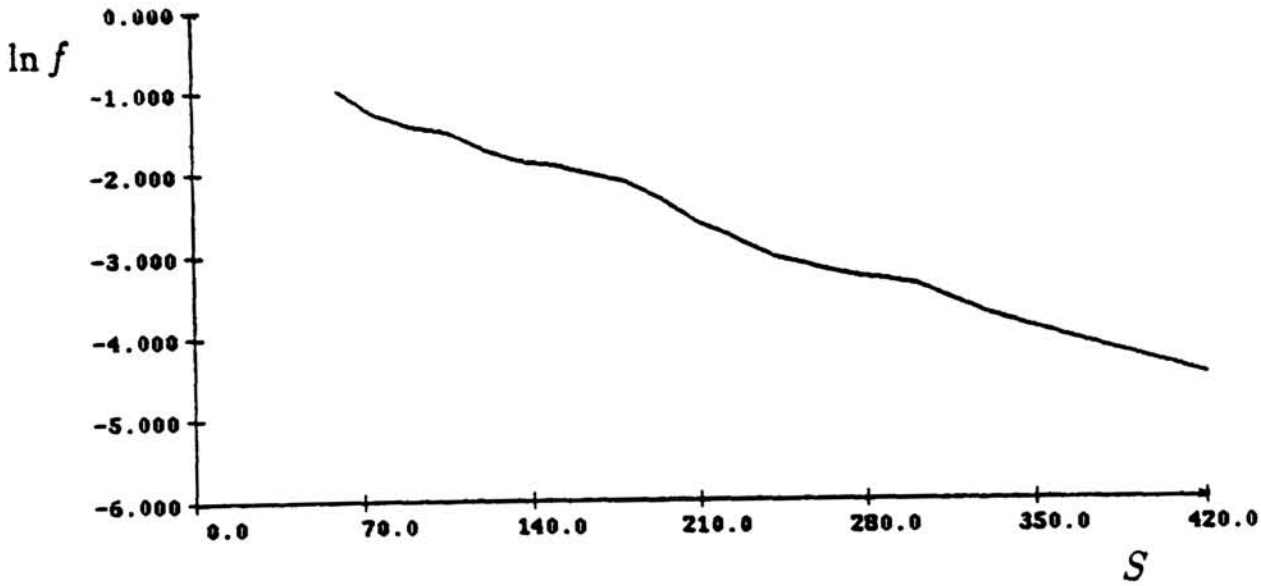

Figure 2: $\ln f$ as a function of $S$. $d = 25$. The slope was $\approx -0.01$

at $S = \alpha d$ for various values of $\alpha$. As Figure 3 shows, the failure rate remains more or less constant for fixed values of $\alpha$, indicating a linear scaling relationship with $d$. Thus $O(d)$ training patterns should be required to learn majority to a fixed level of performance. Note that if we require *perfect* learning, then the failure rate has to be $\leq 1/(2^d - S) \sim 1/2^d$. By substituting this for $f$ in the above formula and solving for $S$, we get that $(\frac{1}{\beta})(d \ln 2 + \ln \alpha)$ patterns are required. The extra factor of $d$ suggests that $O(d^2)$ would be required to learn majority perfectly. We will show in Section 6.1 that this is actually an overestimate.

# 5    THE INPUT REPRESENTATION

So far in our simulations we have used the representation commonly used for boolean predicates. Whenever an input feature has been true, we clamped the corresponding input unit to a 1, and when it has been off we have clamped it to a 0. There is no reason, however, why some other representation couldn't have been used. Notice that in back propagation the weight change is proportional to the incoming input signal, hence the weight from a particular input unit to the output unit is changed only when the pattern is misclassified *and* the input unit is non-zero. The weight remains unchanged when the input unit is 0. If the 0,1 representation were changed to a -1,+1 representation each weight will be changed more often, hence the network should learn the training set quicker (simulations in [Stornetta and Huberman, 87] reported such a decrease in training time using a $-\frac{1}{2}, +\frac{1}{2}$ representation.)

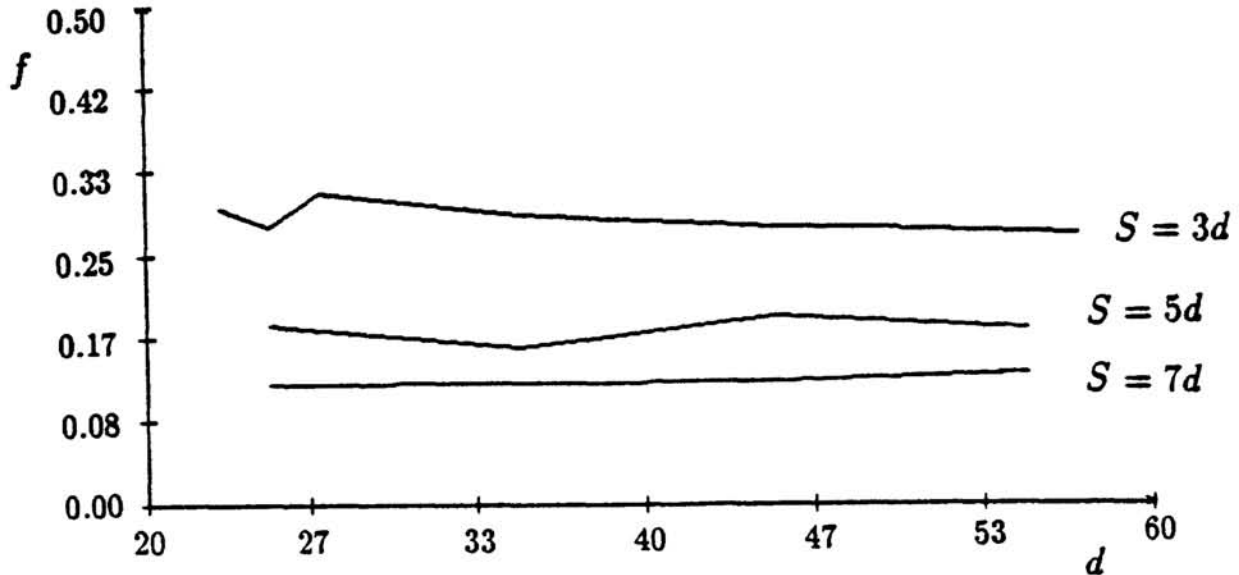

Figure 3: Failure rate vs $d$ with $S = 3d, 5d, 7d$.

We found that not only did the training time decrease with the new representation, the generalization of the network improved significantly. The scaling of the failure rate with respect to $S$ is unchanged, but for any fixed value of $S$, the generalization is about $5 - 10\%$ better. Also, the scaling with respect to $d$ is still linear, but the constant for a fixed performance level is smaller. Although the exact reason for the improved generalization is unclear, the following might be a plausible reason. A weight is changed only if the corresponding input is non-zero. By the definition of the majority function, the average number of units that are on for the positive instances is higher than for the negative instances. Hence, using the 0,1 representation, the weight changes are more pronounced for the positive instances than for the negative instances. Since the weights are changed whenever a pattern is misclassified, the net result is that the weight change is greater when a positive event is misclassified than when a negative event is misclassified. Thus, there seems to be a bias in the 0,1 representation for correcting the hyperplane more when a positive event is misclassified. In the new representation, both positive and negative events are treated equally hence it is unbiased.

The basic lesson here seems to be that one should carefully examine *every* choice that has been made during the design process. The representation of the input, even down to such low level details as deciding whether "off" should be represented as 0 or -1, could make a significant difference in the generalization.

# 6    BORDER PATTERNS

We now consider a method for improving the generalization by intelligently selecting the patterns in the training set. Normally, for a given training set, when the inputs are spread evenly around the input space, there can be several generalizations which are consistent with the patterns. The performance of the network on the test set becomes a random event, depending on the initial state of the network. If practical, it makes sense to choose training patterns which can limit the possible generalizations. In particular, if we can find those examples which are closest to the separating surface, we can maximally constrain the number of generalizations. The solution that the network converges to using these "border" patterns should have a higher probability of being a good separator. In general finding a perfect set of border patterns can be computationally expensive, however there might exist simple heuristics which can help select good training examples.

We explored one heuristic for choosing such points: selecting only those patterns in which the number of 1's is either one less or one more than half the number of input units. Intuitively, these inputs should be close to the desired separating surface, thereby constraining the network more than random patterns would. Our results show that using only border patterns in the training set, there is a large increase in the expected performance of the network for a given $S$. In addition, the scaling behavior as a function of $S$ seems to be very different and is faster than an exponential decrease. (Figure 4 shows typical failure rate vs $S$ curves comparing border patterns, the -1,+1 representation, and the 0,1 representation.)

## 6.1    BORDER PATTERNS AND PERFECT LEARNING

We say the network has *perfectly learned* a function when the test patterns are never misclassified. For the majority function, one can argue that at least some border patterns must be present in order to *guarantee* perfect performance. If no border patterns were in the training set, then the network could have learned the $\frac{d}{2} - 1$ of $d$ or the $\frac{d}{2} + 1$ of $d$ function. Furthermore, if we know that a certain number of border patterns is guaranteed to give perfect performance, say $b(d)$, then given the probability that a random pattern is a border pattern, we can calculate the expected number of random patterns sufficient to learn majority.

For odd $d$, there are $2 * \begin{pmatrix} d \\ \frac{d}{2} \end{pmatrix}$ border patterns, so the probability of choosing a border pattern randomly is:

$$\frac{\begin{pmatrix} d \\ \frac{d}{2} \end{pmatrix}}{2^{d-1}}$$

As $d$ gets larger this probability decreases as $1/\sqrt{d}$.* The expected number of randomly chosen patterns required before $b(d)$ border patterns are chosen is therefore:

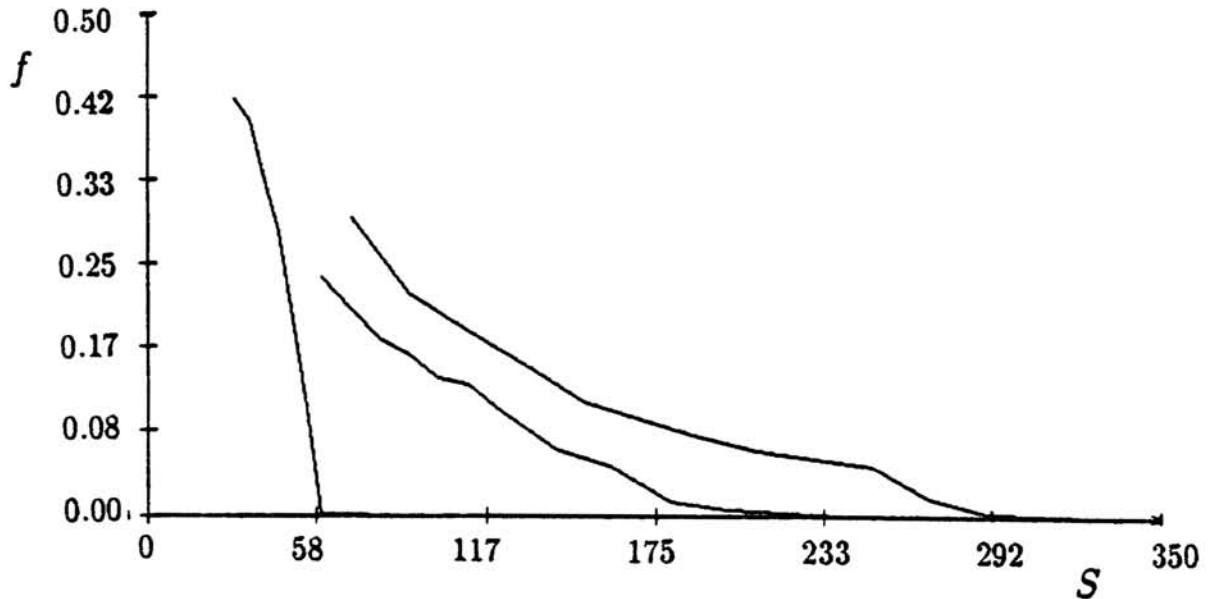

Figure 4: Graph showing the average failure rate vs. $S$ using the 0,1 representation (right), the -1,+1 representation (middle), and using border patterns (left). The network had 23 inputs units and was tested on a test set consisting of 1024 patterns.

$b(d)\sqrt{d}$. From our data we find that $3d$ border patterns are always sufficient to learn the test set perfectly. From this, and from the theoretical results in [Cover, 65], we can be confident that $b(d)$ is linear in $d$. Thus, $O(d^{3/2})$ random patterns should be sufficient to learn majority perfectly.

It should be mentioned that border patterns are not the only patterns which contribute to the generalization of the network. Figure 5 shows that the failure rate of the network when trained with random training patterns which happen to contain $b$ border patterns is substantially better than a training set consisting of only $b$ border patterns. Note that perfect performance is achieved at the same point in both cases.

# 7   CONCLUSION

In this paper we have described a systematic study of some of the various factors affecting scaling and generalization in neural networks. Using empirical studies in a simple test domain, we were able to obtain precise scaling relationships between the performance of the network, the number of training patterns, and the size of the network. It was shown that for a fixed network size, the failure rate decreases exponentially with the size of the training set. The number of patterns required to

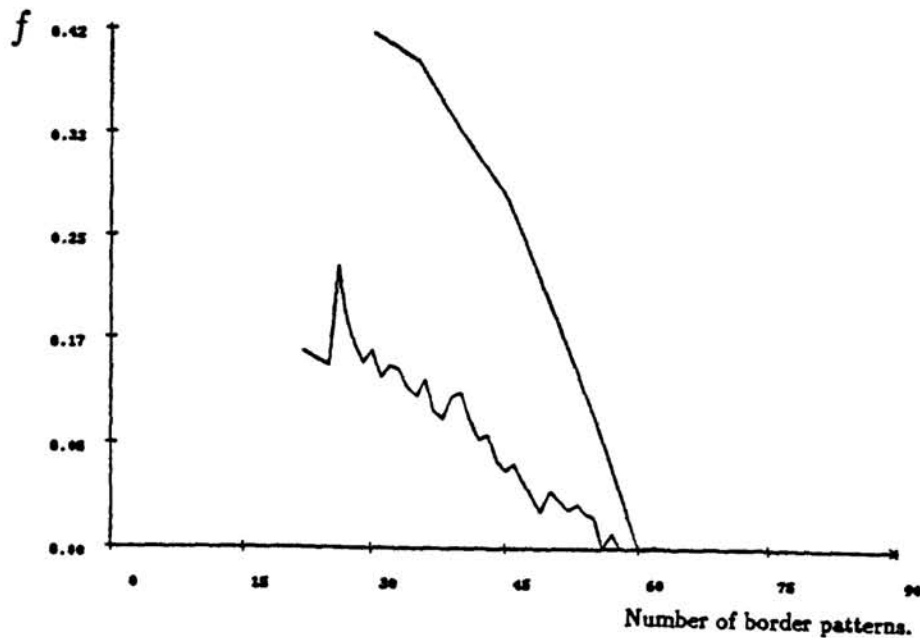

Figure 5: This figure compares the failure rate on a random training set which happens to contain $b$ border patterns (bottom plot) with a training set composed of only $b$ border patterns (top plot).

achieve a fixed performance level was shown to increase linearly with the network size.

A general finding was that the performance of the network was very sensitive to a number of factors. A slight change in the input representation caused a jump in the performance of the network. The specific patterns in the training set had a large influence on the final weights and on the generalization. By selecting the training patterns intelligently, the performance of the network was increased significantly.

The notion of border patterns were introduced as the most interesting patterns in the training set. As far as the number of patterns required to teach a function to the network, these patterns are near optimal. It was shown that a network trained only on border patterns generalizes substantially better than one trained on the same number of random patterns. Border patterns were also used to derive an expected bound on the number of random patterns sufficient to learn majority perfectly. It was shown tha,t on average, $O(d^{3/2})$ random patterns are sufficient to learn majority perfectly.

In conclusion, this paper advocates a careful study of the process of generalization in neural networks. There are a large number of different factors which can affect the performance. Any assumptions made when applying neural networks to a real-world problem should be made with care. Although much more work needs to be

done, it was shown that many of the issues can be effectively studied in a simple test domain.

## Acknowledgements

We thank T. Sejnowski, R. Rivest and A. Barron for helpful discussions. We also thank T. Sejnowski and B. Bogstad for assistance in development of the simulator code. This work was partially supported by the National Center for Supercomputing Applications and by National Science Foundation grant Phy 86-58062.

## Footnotes

[0]* The material contained in this paper is a condensation of portions of the first author's M.S. thesis [Ahmad, 88].

[0]* This can be shown using Stirling's approximation to d!.

# References

[Ahmad, 88] S. Ahmad. *A Study of Scaling and Generalization in Neural Networks.* Technical Report UIUCDCS-R-88-1454, Department of Computer Science, University of Illinois, Urbana-Champaign, IL, 1988.

[Cover, 65] T. Cover. Geometric and satistical properties of systems of linear equations. *IEEE Trans. Elect. Comp.*, 14:326–334, 1965.

[Minsky and Papert, 69] Marvin Minsky and Seymour Papert. *Perceptrons.* MIT Press, Cambridge, Mass., 1969.

[Muroga, 71] S Muroga. *Threshold Logic and its Applications.* Wiley, New York, 1971.

[Rumelhart and McClelland, 86] D. E. Rumelhart and J. L. McClelland, editors. *Parallel Distributed Processing: Explorations in the Microstructure of Cognition: Foundations.* Volume 1, MIT Press, Cambridge, Mass., 1986.

[Stornetta and Huberman, 87] W.S. Stornetta and B.A. Huberman. An improved three-layer, back propagation algorithm. In *Proceedings of the IEEE First International Conference on Neural Networks*, San Diego, CA, 1987.

[Sejnowski and Rosenberg, 87] T.J. Sejnowski and C.R. Rosenberg. Parallel networks that learn to pronounce English text. *Complex Systems*, 1:145–168, 1987.

[Tesauro and Sejnowski, 88] G. Tesauro and T.J. Sejnowski. *A Parallel Network that Learns to Play Backgammon.* Technical Report CCSR-88-2, Center for Complex Systems Research, University of Illinois, Urbana-Champaign, IL, 1988.